# Modeling human function learning
# with Gaussian processes

**Thomas L. Griffiths   Christopher G. Lucas   Joseph J. Williams**
Department of Psychology
University of California, Berkeley
Berkeley, CA 94720-1650
{tom_griffiths,clucas,joseph_williams}@berkeley.edu

**Michael L. Kalish**
Institute of Cognitive Science
University of Louisiana at Lafayette
Lafayette, LA 70504-3772
kalish@lousiana.edu

## Abstract

Accounts of how people learn functional relationships between continuous variables have tended to focus on two possibilities: that people are estimating explicit functions, or that they are performing associative learning supported by similarity. We provide a rational analysis of function learning, drawing on work on regression in machine learning and statistics. Using the equivalence of Bayesian linear regression and Gaussian processes, we show that learning explicit rules and using similarity can be seen as two views of one solution to this problem. We use this insight to define a Gaussian process model of human function learning that combines the strengths of both approaches.

## 1   Introduction

Much research on how people acquire knowledge focuses on discrete structures, such as the nature of categories or the existence of causal relationships. However, our knowledge of the world also includes relationships between continuous variables, such as the difference between linear and exponential growth, or the form of causal relationships, such as how pressing the accelerator of a car influences its velocity. Research on how people learn relationships between two continuous variables – known in the psychological literature as *function learning* – has tended to emphasize two different ways in which people could be solving this problem. One class of theories (e.g., [1, 2, 3]) suggests that people are learning an explicit function from a given class, such as the polynomials of degree $k$. This approach attributes rich representations to human learners, but has traditionally given limited treatment to the question of how such representations could be acquired. A second approach (e.g., [4, 5]) emphasizes the possibility that people learn by forming associations between observed values of input and output variables, and generalize based on the similarity of new inputs to old. This approach has a clear account of the underlying learning mechanisms, but faces challenges in explaining how people generalize so broadly beyond their experience, making predictions about variable values that are significantly removed from their previous observations. Most recently, hybrids of these two approaches have been proposed (e.g., [6, 7]), with explicit functions being represented, but associative learning.

Previous models of human function learning have been oriented towards understanding the psychological processes by which people solve this problem. In this paper, we take a different approach,

presenting a rational analysis of function learning, in the spirit of [8]. This rational analysis provides a way to understand the relationship between the two approaches that have dominated previous work – rules and similarity – and suggests how they might be combined. The basic strategy we pursue is to consider the abstract computational problem involved in function learning, and then to explore optimal solutions to that problem with the goal of shedding light on human behavior. In particular, the problem of learning a functional relationship between two continuous variables is an instance of *regression*, and has been extensively studied in machine learning and statistics.

There are a variety of solution to regression problems, but we focus on methods related to Bayesian linear regression (e.g., [9]), which allow us to make the expectations of learners about the form of functions explicit through a prior distribution. Bayesian linear regression is also directly related to a nonparametric approach known as Gaussian process prediction (e.g., [10]), in which predictions about the values of an output variable are based on the similarity between values of an input variable. We use this relationship to connect the two traditional approaches to modeling function learning, as it shows that learning rules that describe functions and specifying the similarity between stimuli for use in associative learning are not mutually exclusive alternatives, but rather two views of the same solution to this problem. We exploit this fact to define a rational model of human function learning that incorporates the strengths of both approaches.

## 2 Models of human function learning

In this section we review the two traditional approaches to modeling human function learning – rules and similarity – and some more recent hybrid approaches that combine the two.

### 2.1 Representing functions with rules

The idea that people might represent functions explicitly appears in one of the first papers on human function learning [1]. This paper proposed that people assume a particular class of functions (such as polynomials of degree $k$) and use the available observations to estimate the parameters of those functions, forming a representation that goes beyond the observed values of the variables involved. Consistent with this hypothesis, people learned linear and quadratic functions better than random pairings of values for two variables, and extrapolated appropriately. Similar assumptions guided subsequent work exploring the ease with which people learn functions from different classes (e.g., [2], and papers have tested statistical regression schemes as potential models of learning, examining how well human responses were described by different forms of nonlinear regression (e.g., [3]).

### 2.2 Similarity and associative learning

Associative learning models propose that people do not learn relationships between continuous variables by explicitly learning rules, but by forging associations between observed variable pairs and generalizing based on the similarity of new variable values to old. The first model to implement this approach was the Associative-Learning Model (ALM; [4, 5]), in which input and output arrays are used to represent a range of values for the two variables between which the functional relationship holds. Presentation of an input activates input nodes close to that value, with activation falling off as a Gaussian function of distance, explicitly implementing a theory of similarity in the input space. Learned weights determine the activation of the output nodes, being a weighted linear function of the activation of the input nodes. Associative learning for the weights is performed by applying gradient descent on the squared error between current output activation and the correct value. In practice, this approach performs well when interpolating between observed values, but poorly when extrapolating beyond those values. As a consequence, the same authors introduced the Extrapolation-Association Model (EXAM), which constructs a linear approximation to the output of the ALM when selecting responses, producing a bias towards linearity that better matches human judgments.

### 2.3 Hybrid approaches

Several papers have explored methods for combining rule-like representations of functions with associative learning. One example of such an approach is the set of rule-based models explored in [6]. These models used the same kind of input representation as ALM and EXAM, with activation

of a set of nodes similar to the input value. However, the models also feature a set of hidden units, where each hidden unit corresponds to a different parameterization of a rule from a given class (polynomial, Fourier, or logistic). The values of the hidden nodes – corresponding to the values of the rules they instantiate – are combined linearly to obtain output predictions, with the weight of each hidden node being learned through gradient descent (with a penalty for the curvature of the functions involved). A more complex instance of this kind of approach is the Population of Linear Experts (POLE) model [7], in which hidden units each represent different linear functions, but the weights from input to hidden nodes indicate which linear function should be used to make predictions for particular input values. As a consequence, the model can learn non-linear functions by identifying a series of local linear approximations, and can even model situations in which people seem to learn different functions in different parts of the input space.

## 3 Rational solutions to regression problems

The models outlined in the previous section all aim to describe the psychological processes involved in human function learning. In this section, we consider the abstract computational problem underlying this task, using optimal solutions to this problem to shed light on both previous models and human learning. Viewed abstractly, the computational problem behind function learning is to learn a function $f$ mapping from $x$ to $y$ from a set of real-valued observations $\mathbf{x}_n = (x_1, \ldots, x_n)$ and $\mathbf{t}_n = (t_1, \ldots, t_n)$, where $t_i$ is assumed to be the true value $y_i = f(x_i)$ obscured by additive noise.[1] In machine learning and statistics, this is referred to as a *regression* problem. In this section, we discuss how this problem can be solved using Bayesian statistics, and how the result of this approach is related to Gaussian processes. Our presentation follows that in [10].

### 3.1 Bayesian linear regression

Ideally, we would seek to solve our regression problem by combining some prior beliefs about the probability of encountering different kinds of functions in the world with the information provided by $\mathbf{x}$ and $\mathbf{t}$. We can do this by applying Bayes' rule, with

$$p(f|\mathbf{x}_n, \mathbf{t}_n) = \frac{p(\mathbf{t}_n|f, \mathbf{x}_n)p(f)}{\int_{\mathcal{F}} p(\mathbf{t}_n|f, \mathbf{x}_n)p(f)\,df}, \tag{1}$$

where $p(f)$ is the *prior* distribution over functions in the hypothesis space $\mathcal{F}$, $p(\mathbf{t}_n|f, \mathbf{x}_n)$ is the probability of observing the values of $\mathbf{t}_n$ if $f$ were the true function, known as the *likelihood*, and $p(f|\mathbf{x}_n, \mathbf{t}_n)$ is the *posterior* distribution over functions given the observations $\mathbf{x}_n$ and $\mathbf{t}_n$. In many cases, the likelihood is defined by assuming that the values of $t_i$ are independent given $f$ and $x_i$, being Gaussian with mean $y_i = f(x_i)$ and variance $\sigma_t^2$. Predictions about the value of the function $f$ for a new input $x_{n+1}$ can be made by integrating over the posterior distribution,

$$p(y_{n+1}|x_{n+1}, \mathbf{t}_n, \mathbf{x}_n) = \int_f p(y_{n+1}|f, x_{n+1})p(f|\mathbf{x}_n, \mathbf{t}_n)\,df, \tag{2}$$

where $p(y_{n+1}|f, x_{n+1})$ is a delta function placing all of its mass on $y_{n+1} = f(x_{n+1})$.

Performing the calculations outlined in the previous paragraph for a general hypothesis space $\mathcal{F}$ is challenging, but becomes straightforward if we limit the hypothesis space to certain specific classes of functions. If we take $\mathcal{F}$ to be all linear functions of the form $y = b_0 + x b_1$, then our problem takes the familiar form of linear regression. To perform Bayesian linear regression, we need to define a prior $p(f)$ over all linear functions. Since these functions are identified by the parameters $b_0$ and $b_1$, it is sufficient to define a prior over $\mathbf{b} = (b_0, b_1)$, which we can do by assuming that $\mathbf{b}$ follows a multivariate Gaussian distribution with mean zero and covariance $\boldsymbol{\Sigma}_b$. Applying Equation 1 then results in a multivariate Gaussian posterior distribution on $\mathbf{b}$ (see [9]) with

$$E[\mathbf{b}|\mathbf{x}_n, \mathbf{t}_n] = \left(\sigma_t^2 \boldsymbol{\Sigma}_b^{-1} + \mathbf{X}_n^T \mathbf{X}_n\right)^{-1} \mathbf{X}_n^T \mathbf{t}_n \tag{3}$$

$$\mathrm{cov}[\mathbf{b}|\mathbf{x}_n, \mathbf{y}_n] = \left(\boldsymbol{\Sigma}_b^{-1} + \frac{1}{\sigma_t^2} \mathbf{X}_n^T \mathbf{X}_n\right)^{-1} \tag{4}$$

where $\mathbf{X}_n = [\mathbf{1}_n \ \mathbf{x}_n]$ (ie. a matrix with a vector of ones horizontally concatenated with $\mathbf{x}_{n+1}$) Since $y_{n+1}$ is simply a linear function of $\mathbf{b}$, applying Equation 2 yields a Gaussian predictive distribution, with $y_{n+1}$ having mean $[1 \ x_{n+1}]E[\mathbf{b}|\mathbf{x}_n, \mathbf{t}_n]$ and variance $[1 \ x_{n+1}]\text{cov}[\mathbf{b}|\mathbf{x}_n, \mathbf{t}_n][1 \ x_{n+1}]^T$. The predictive distribution for $t_{n+1}$ is similar, but with the addition of $\sigma_t^2$ to the variance.

While considering only linear functions might seem overly restrictive, linear regression actually gives us the basic tools we need to solve this problem for more general classes of functions. Many classes of functions can be described as linear combinations of a small set of basis functions. For example, all $k$th degree polynomials are linear combinations of functions of the form 1 (the constant function), $x$, $x^2$, ..., $x^k$. Letting $\phi^{(1)}, \ldots, \phi^{(k)}$ denote a set of functions, we can define a prior on the class of functions that are linear combinations of this basis by expressing such functions in the form $f(x) = b_0 + \phi^{(1)}(x)b_1 + \ldots + \phi^{(k)}(x)b_k$ and defining a prior on the vector of weights $\mathbf{b}$. If we take the prior to be Gaussian, we reach the same solution as outlined in the previous paragraph, substituting $\mathbf{\Phi} = [\mathbf{1}_n \ \phi^{(1)}(\mathbf{x}_n) \ \ldots \ \phi^{(k)}(\mathbf{x}_n)]$ for $\mathbf{X}$ and $[1 \ \phi^{(1)}(x_{n+1}) \ \ldots \ \phi^{(k)}(x_{n+1})]$ for $[1 \ x_{n+1}]$, where $\phi(\mathbf{x}_n) = [\phi(x_1) \ \ldots \ \phi(x_n)]^T$.

## 3.2 Gaussian processes

If our goal were merely to predict $y_{n+1}$ from $x_{n+1}$, $\mathbf{y}_n$, and $\mathbf{x}_n$, we might consider a different approach, simply defining a joint distribution on $\mathbf{y}_{n+1}$ given $\mathbf{x}_{n+1}$ and conditioning on $\mathbf{y}_n$. For example, we might take the $\mathbf{y}_{n+1}$ to be jointly Gaussian, with covariance matrix

$$\mathbf{K}_{n+1} = \begin{pmatrix} \mathbf{K}_n & \mathbf{k}_{n,n+1} \\ \mathbf{k}_{n,n+1}^T & k_{n+1} \end{pmatrix} \tag{5}$$

where $\mathbf{K}_n$ depends on the values of $\mathbf{x}_n$, $\mathbf{k}_{n,n+1}$ depends on $\mathbf{x}_n$ and $x_{n+1}$, and $k_{n+1}$ depends only on $x_{n+1}$. If we condition on $\mathbf{y}_n$, the distribution of $y_{n+1}$ is Gaussian with mean $\mathbf{k}_{n,n+1}^T \mathbf{K}_n^{-1}\mathbf{y}$ and variance $k_{n+1} - \mathbf{k}_{n,n+1}^T \mathbf{K}_n^{-1}\mathbf{k}_{n,n+1}$. This approach to prediction uses a *Gaussian process*, a stochastic process that induces a Gaussian distribution on $\mathbf{y}$ based on the values of $\mathbf{x}$. This approach can also be extended to allow us to predict $y_{n+1}$ from $x_{n+1}$, $\mathbf{t}_n$, and $\mathbf{x}_n$ by adding $\sigma_t^2 \mathbf{I}_n$ to $\mathbf{K}_n$, where $\mathbf{I}_n$ is the $n \times n$ identity matrix, to take into account the additional variance associated with $\mathbf{t}_n$.

The covariance matrix $\mathbf{K}_{n+1}$ is specified using a two-place function in $x$ known as a *kernel*, with $K_{ij} = K(x_i, x_j)$. Any kernel that results in an appropriate (symmetric, positive-definite) covariance matrix for all $\mathbf{x}$ can be used. Common kinds of kernels include radial basis functions, e.g.,

$$K(x_i, x_j) = \theta_1^2 \exp(-\frac{1}{\theta_2^2}(x_i - x_j)^2) \tag{6}$$

with values of $y$ for which values of $x$ are close being correlated, and periodic functions, e.g.,

$$K(x_i, x_j) = \theta_3^2 \exp(\theta_4^2(\cos(\frac{2\pi}{\theta_5}[x_i - x_j]))) \tag{7}$$

indicating that values of $y$ for which values of $x$ are close relative to the period $\theta_3$ are likely to be highly correlated. Gaussian processes thus provide a flexible approach to prediction, with the kernel defining which values of $x$ are likely to have similar values of $y$.

## 3.3 Two views of regression

Bayesian linear regression and Gaussian processes appear to be quite different approaches. In Bayesian linear regression, a hypothesis space of functions is identified, a prior on that space is defined, and predictions are formed averaging over the posterior, while Gaussian processes simply use the similarity between different values of $x$, as expressed through a kernel, to predict correlations in values of $y$. It might thus come as a surprise that these approaches are equivalent.

Showing that Bayesian linear regression corresponds to Gaussian process prediction is straightforward. The assumption of linearity means that the vector $\mathbf{y}_{n+1}$ is equal to $\mathbf{X}_{n+1}\mathbf{b}$. It follows that $p(\mathbf{y}_{n+1}|\mathbf{x}_{n+1})$ is a multivariate Gaussian distribution with mean zero and covariance matrix $\mathbf{X}_{n+1}\mathbf{\Sigma}_b\mathbf{X}_{n+1}^T$. Bayesian linear regression thus corresponds to prediction using Gaussian processes, with this covariance matrix playing the role of $\mathbf{K}_{n+1}$ above (ie. using the kernel function $K(x_i, x_j) = [1 \ x_i][1 \ x_j]^T$). Using a richer set of basis functions corresponds to taking $\mathbf{K}_{n+1} = \mathbf{\Phi}_{n+1}\mathbf{\Sigma}_b\mathbf{\Phi}_{n+1}^T$ (ie. $K(x_i, x_j) = [1 \ \phi^{(1)}(x_i) \ \ldots \ \phi^{(k)}(x_i)][1 \ \phi^{(1)}(x_i) \ \ldots \ \phi^{(k)}(x_i)]^T$).

It is also possible to show that Gaussian process prediction can always be interpreted as Bayesian linear regression, albeit with potentially infinitely many basis functions. Just as we can express a covariance matrix in terms of its eigenvectors and eigenvalues, we can express a given kernel $K(x_i, x_j)$ in terms of its eigenfunctions $\phi$ and eigenvalues $\boldsymbol{\lambda}$, with

$$K(x_i, x_j) = \sum_{k=1}^{\infty} \lambda_k \phi^{(k)}(x_i) \phi^{(k)}(x_j) \tag{8}$$

for any $x_i$ and $x_j$. Using the results from the previous paragraph, any kernel can be viewed as the result of performing Bayesian linear regression with a set of basis functions corresponding to its eigenfunctions, and a prior with covariance matrix $\boldsymbol{\Sigma}_b = \mathrm{diag}(\boldsymbol{\lambda})$.

These results establish an important duality between Bayesian linear regression and Gaussian processes: for every prior on functions, there exists a corresponding kernel, and for every kernel, there exists a corresponding prior on functions. Bayesian linear regression and prediction with Gaussian processes are thus just two views of the same solution to regression problems.

## 4 Combining rules and similarity through Gaussian processes

The results outlined in the previous section suggest that learning rules and generalizing based on similarity should not be viewed as conflicting accounts of human function learning. In this section, we briefly highlight how previous accounts of function learning connect to statistical models, and then use this insight to define a model that combines the strengths of both approaches.

### 4.1 Reinterpreting previous accounts of human function learning

The models presented above were chosen because the contrast between rules and similarity in function learning is analogous to the difference between Bayesian linear regression and Gaussian processes. The idea that human function learning can be viewed as a kind of statistical regression [1, 3] clearly connects directly to Bayesian linear regression. While there is no direct formal correspondence, the basic ideas behind Gaussian process regression with a radial basis kernel and similarity-based models such as ALM are closely related. In particular, ALM has many commonalities with radial-basis function neural networks, which are directly related to Gaussian processes [11]. Gaussian processes with radial-basis kernels can thus be viewed as implementing a simple kind of similarity-based generalization, predicting similar $y$ values for stimuli with similar $x$ values. Finally, the hybrid approach to rule learning taken in [6] is also closely related to Bayesian linear regression. The rules represented by the hidden units serve as a basis set that specify a class of functions, and applying penalized gradient descent on the weights assigned to those basis elements serves as an online algorithm for finding the function with highest posterior probability [12].

### 4.2 Mixing functions in a Gaussian process model

The relationship between Gaussian processes and Bayesian linear regression suggests that we can define a single model that exploits both similarity and rules in forming predictions. In particular, we can do this by taking a prior that covers a broad class of functions – including those consistent with a radial basis kernel – or, equivalently, modeling $\mathbf{y}$ as being produced by a Gaussian process with a kernel corresponding to one of a small number of types. Specifically, we assume that observations are generated by choosing a type of function from the set $\{\text{Positive Linear}, \text{Negative Linear}, \text{Quadratic}, \text{Nonlinear}\}$, where the probabilities of these alternatives are defined by the vector $\boldsymbol{\pi}$, and then sampling $\mathbf{y}$ from a Gaussian process with a kernel corresponding to the appropriate class of functions. The relevant kernels are introduced in the previous sections (taking "Nonlinear" to correspond to the radial basis kernel), with the "Positive Linear" and "Negative Linear" kernels being derived in a similar way to the standard linear kernel but with the mean of the prior on $\mathbf{b}$ being $[0\ 1]$ and $[1\ -1]$ rather than simply zero.

Using this Gaussian process model allows a learner to make an inference about the type of function from which their observations are drawn, as well as the properties of the function of that type. In practice, we perform probabilistic inference using a Markov chain Monte Carlo (MCMC) algorithm (see [13] for an introduction). This algorithm defines a Markov chain for which the stationary

distribution is the distribution from which we wish to sample. In our case, this is the posterior distribution over types and the hyperparameters for the kernels $\theta$ given the observations $\mathbf{x}$ and $\mathbf{t}$. The hyperparameters include $\theta_1$ and $\theta_2$ defined above and the noise in the observations $\sigma_t^2$. Our MCMC algorithm repeats two steps. The first step is sampling the type of function conditioned on $\mathbf{x}$, $\mathbf{t}$, and the current value of $\theta$, with the probability of each type being proportional to the product of $p(\mathbf{t}_n|\mathbf{x}_n)$ for the corresponding Gaussian process and the prior probability of that type as given by $\pi$. The second step is sampling the value of $\theta$ given $\mathbf{x}_n$, $\mathbf{t}_n$, and the current type, which is done using a Metropolis-Hastings procedure (see [13]), proposing a value for $\theta$ from a Gaussian distribution centered on the current value and deciding whether to accept that value based on the product of the probability it assigns to $\mathbf{t}_n$ given $\mathbf{x}_n$ and the prior $p(\theta)$. We use an uninformative prior on $\theta$.

## 5 Testing the Gaussian process model

Following a recent review of computational models of function learning [6], we look at two quantitative tests of Gaussian processes as an account of human function learning: reproducing the order of difficulty of learning functions of different types, and extrapolation performance. As indicated earlier, there is a large literature consisting of both models and data concerning human function learning, and these simulations are intended to demonstrate the potential of the Gaussian process model rather than to provide an exhaustive test of its performance.

### 5.1 Difficulty of learning

A necessary criterion for a theory of human function learning is accounting for which functions people learn readily and which they find difficult – the relative difficulty of learning various functions. Table 1 is an augmented version of results presented in [6] which compared several models to the empirically observed difficulty of learning a range of functions. Each entry in the table is the mean absolute deviation (MAD) of human or model responses from the actual value of the function, evaluated over the stimuli presented in training. The MAD provides a measure of how difficult it is for people or a given model to learn a function. The data reported for each set of studies are ordered by increasing MAD (corresponding to increasing difficulty). In addition to reproducing the MAD for the models in [6], the table includes results for seven Gaussian process (GP) models.

The seven GP models incorporated different kernel functions by adjusting their prior probability. Drawing on the {Positive Linear, Negative Linear, Quadratic, Nonlinear} set of kernel functions, the most comprehensive model took $\pi = (0.5, 0.4, 0.09, 0.01)$.[2] Six other GP models were examined by assigning certain kernel functions zero prior probability and re-normalizing the modified value of $\pi$ so that the prior probabilities summed to one. The seven distinct GP models are presented in Table 1 and labeled by the kernel functions with non-zero prior probability: Linear (Positive Linear and Negative Linear), Quadratic, Nonlinear (Radial Basis Function), Linear and Quadratic, Linear and Nonlinear, Quadratic and Nonlinear, and Linear, Quadratic, and Nonlinear. The last two rows of Table 1 give the correlations between human and model performance across functions, expressing quantitatively how well each model captured the pattern of human function learning behavior. The GP models perform well according to this metric, providing a closer match to the human data than any of the models considered in [6], with the quadratic kernel and the models with a mixture of kernels tending to provide a closer match to human behavior.

### 5.2 Extrapolation performance

Predicting and explaining people's capacity for generalization – from stimulus-response pairs to judgments about a functional relationship between variables – is the second key component of our account. This capacity is assessed in the way in which people extrapolate, making judgments about stimuli they have not encountered before. Figure 1 shows mean human predictions for a linear, exponential, and quadratic function (from [4]), together with the predictions of the most comprehensive GP model (with Linear, Quadratic and Nonlinear kernel functions). The regions to the left and right of the vertical lines represent extrapolation regions, being input values for which neither people nor

| | | | Hybrid models | | | Gaussian process models | | | | | | |
|---|---|---|---|---|---|---|---|---|---|---|---|---|
| Function | Human | ALM | Poly | Fourier | Logistic | Linear | Quad | RBF | LQ | LR | QR | LQR |
| Byun (1995, Expt 1B) | | | | | | | | | | | | |
|   Linear | .20 | .04 | .04 | .05 | .16 | .0002 | .004 | .06 | .0002 | .0002 | .001 | .0001 |
|   Square root | .35 | .05 | .06 | .06 | .19 | .06 | .02 | .05 | .02 | .03 | .02 | .02 |
| Byun (1995, Expt 1A) | | | | | | | | | | | | |
|   Linear | .15 | .10 | .33 | .33 | .17 | .0003 | .004 | .04 | .0002 | .0002 | .0009 | .0001 |
|   Power, pos. acc. | .20 | .12 | .37 | .37 | .24 | .11 | .004 | .08 | .004 | .05 | .003 | .003 |
|   Power, neg. acc. | .23 | .12 | .36 | .36 | .19 | .06 | .02 | .05 | .02 | .03 | .02 | .02 |
|   Logarithmic | .30 | .14 | .41 | .41 | .19 | .10 | .04 | .07 | .04 | .05 | .03 | .03 |
|   Logistic | .39 | .18 | .51 | .52 | .33 | .20 | .20 | .22 | .20 | .18 | .18 | .18 |
| Byun (1995, Expt 2) | | | | | | | | | | | | |
|   Linear | .18 | .01 | .18 | .19 | .12 | .0003 | .005 | .05 | .0003 | .0002 | .001 | .0002 |
|   Quadratic | .28 | .03 | .31 | .31 | .24 | .20 | .09 | .14 | .09 | .12 | .04 | .04 |
|   Cyclic | .68 | .32 | .41 | .40 | .68 | .50 | .50 | .50 | .50 | .49 | .49 | .49 |
| Delosh, Busemeyer, & McDaniel (1997) | | | | | | | | | | | | |
|   Linear | .10 | .04 | .11 | .11 | .04 | .0005 | .005 | .03 | .0005 | .0003 | .002 | .0004 |
|   Exponential | .15 | .05 | .17 | .17 | .02 | .03 | .01 | .02 | .01 | .02 | .009 | .01 |
|   Quadratic | .24 | .07 | .27 | .27 | .11 | .1 | .06 | .07 | .06 | .06 | .04 | .04 |
| Correlation of human and model performance | | | | | | | | | | | | |
|   Linear | 1.0 | .83 | .45 | .45 | .93 | .93 | .92 | .92 | .93 | .93 | .92 | .92 |
|   Rank-order | 1.0 | .55 | .51 | .51 | .77 | .76 | .80 | .75 | .83 | .83 | .82 | .83 |

Table 1: Difficulty of learning results. Rows correspond to functions learned in experiments reviewed in [6]. Columns give the mean absolute deviation (MAD) from the true functions for human learners and different models (Gaussian process models with multiple kernels are denoted by the initials of their kernels, e.g., LQR = Linear, Quadratic, and Radial Basis Function). Human MAD values represent sample means (for a single subject over trials, then over subjects), and reflect both estimation and production errors, being higher than model MAD values which are computed using deterministic model predictions and thus reflect only estimation error. The last two rows give the linear and rank-order correlations of the human and model MAD values, providing an indication of how well the model matches the difficulty people have in learning different functions.

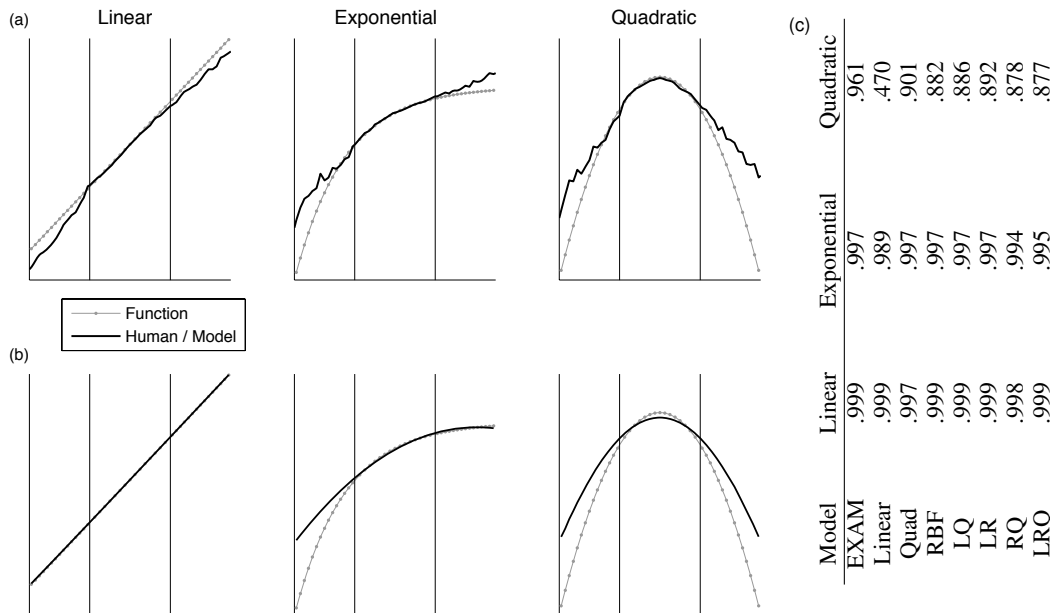

Figure 1: Extrapolation performance. (a)-(b) Mean predictions on linear, exponential, and quadratic functions for (a) human participants (from [4]) and (b) a Gaussian process model with Linear, Quadratic, and Nonlinear kernels. Training data were presented in the region between the vertical lines, and extrapolation performance was evaluated outside this region. (c) Correlations between human and model extrapolation. Gaussian process models are denoted as in Table 1.

the model were trained. Both people and the model extrapolate near optimally on the linear function, and reasonably accurate extrapolation also occurs for the exponential and quadratic function. However, there is a bias towards a linear slope in the extrapolation of the exponential and quadratic functions, with extreme values of the quadratic and exponential function being overestimated.

Quantitative measures of extrapolation performance are shown in Figure 1 (c), which gives the correlation between human and model predictions for EXAM [4, 5] and the seven GP models. While none of the GP models produce quite as high a correlation as EXAM on all three functions, all of the models except that with just the linear kernel produce respectable correlations. It is particularly notable that this performance is achieved without the optimization of any free parameters, while the predictions of EXAM were the result of optimizing two parameters for each of the three functions.

## 6 Conclusions

We have presented a rational account of human function learning, drawing on ideas from machine learning and statistics to show that the two approaches that have dominated previous work – rules and similarity – can be interpreted as two views of the same kind of optimal solution to this problem. Our Gaussian process model combines the strengths of both approaches, using a mixture of kernels to allow systematic extrapolation as well as sensitive non-linear interpolation. Tests of the performance of this model on benchmark datasets show that it can capture some of the basic phenomena of human function learning, and is competitive with existing process models. In future work, we aim to extend this Gaussian process model to allow it to produce some of the more complex phenomena of human function learning, such as non-monotonic extrapolation (via periodic kernels) and learning different functions in different parts of the input space (via mixture modeling).

**Acknowledgments.** This work was supported by grant FA9550-07-1-0351 from the Air Force Office of Scientific Research and grants 0704034 and 0544705 from the National Science Foundation.

## Footnotes

[1]Following much of the literature on human function learning, we consider only one-dimensional functions, but this approach generalizes naturally to the multi-dimensional case.

[2]The selection of these values was guided by results indicating the order of difficulty of learning functions of these different types for human learners, but we did not optimize $\pi$ with respect to the criteria reported here.

## References

[1] J. D. Carroll. *Functional learning: The learning of continuous functional mappings relating stimulus and response continua*. Education Testing Service, Princeton, NJ, 1963.

[2] B. Brehmer. Hypotheses about relations between scaled variables in the learning of probabilistic inference tasks. *Organizational Behavior and Human Decision Processes*, 11:1–27, 1974.

[3] K. Koh and D. E. Meyer. Function learning: Induction of continuous stimulus-response relations. *Journal of Experimental Psychology: Learning, Memory, and Cognition*, 17:811–836, 1991.

[4] E. L. DeLosh, J. R. Busemeyer, and M. A. McDaniel. Extrapolation: The sine qua non of abstraction in function learning. *Journal of Experimental Psychology: Learning, Memory, and Cognition*, 23:968–986, 1997.

[5] J. R. Busemeyer, E. Byun, E. L. DeLosh, and M. A. McDaniel. Learning functional relations based on experience with input-output pairs by humans and artificial neural networks. In K. Lamberts and D. Shanks, editors, *Concepts and Categories*, pages 405–437. MIT Press, Cambridge, 1997.

[6] M. A. McDaniel and J. R. Busemeyer. The conceptual basis of function learning and extrapolation: Comparison of rule-based and associative-based models. *Psychonomic Bulletin and Review*, 12:24–42, 2005.

[7] M. Kalish, S. Lewandowsky, and J. Kruschke. Population of linear experts: Knowledge partitioning and function learning. *Psychological Review*, 111:1072–1099, 2004.

[8] J. R. Anderson. *The adaptive character of thought*. Erlbaum, Hillsdale, NJ, 1990.

[9] J. M. Bernardo and A. F. M. Smith. *Bayesian theory*. Wiley, New York, 1994.

[10] C. K. I. Williams. Prediction with Gaussian processes: From linear regression to linear prediction and beyond. In M. I. Jordan, editor, *Learning in Graphical Models*, pages 599–621. MIT Press, Cambridge, MA, 1998.

[11] R. M. Neal. Priors for infinite networks. Technical Report CRG-TR-94-1, Department of Computer Science, University of Toronto, 1994.

[12] D.J.C. MacKay. Probable networks and plausible predictions - a review of practical bayesian methods for supervised neural networks. *Network: Computation in Neural Systems*, 6:469–505, 1995.

[13] W.R. Gilks, S. Richardson, and D. J. Spiegelhalter, editors. *Markov Chain Monte Carlo in Practice*. Chapman and Hall, Suffolk, UK, 1996.

